# Fast subtree kernels on graphs

**Nino Shervashidze, Karsten M. Borgwardt**
Interdepartmental Bioinformatics Group
Max Planck Institutes Tübingen, Germany
{nino.shervashidze,karsten.borgwardt}@tuebingen.mpg.de

## Abstract

In this article, we propose fast subtree kernels on graphs. On graphs with $n$ nodes and $m$ edges and maximum degree $d$, these kernels comparing subtrees of height $h$ can be computed in $O(mh)$, whereas the classic subtree kernel by Ramon & Gärtner scales as $O(n^2 4^d h)$. Key to this efficiency is the observation that the Weisfeiler-Lehman test of isomorphism from graph theory elegantly computes a subtree kernel as a byproduct. Our fast subtree kernels can deal with labeled graphs, scale up easily to large graphs and outperform state-of-the-art graph kernels on several classification benchmark datasets in terms of accuracy and runtime.

## 1 Introduction

Graph kernels have recently evolved into a branch of kernel machines that reaches deep into graph mining. Several different graph kernels have been defined in machine learning which can be categorized into three classes: graph kernels based on walks [5, 7] and paths [2], graph kernels based on limited-size subgraphs [6, 11], and graph kernels based on subtree patterns [9, 10].

While fast computation techniques have been developed for graph kernels based on walks [12] and on limited-size subgraphs [11], it is unclear how to compute subtree kernels efficiently. As a consequence, they have been applied to relatively small graphs representing chemical compounds [9] or handwritten digits [1], with approximately twenty nodes on average. But could one speed up subtree kernels to make them usable on graphs with hundreds of nodes, as they arise in protein structure models or in program flow graphs?

It is a general limitation of graph kernels that they scale poorly to large, labeled graphs with more than 100 nodes. While the efficient kernel computation strategies from [11, 12] are able to compare unlabeled graphs efficiently, the efficient comparison of large, labeled graphs remains an unsolved challenge. Could one speed up subtree kernels to make them the kernel of choice for comparing large, labeled graphs?

The goal of this article is to address both of the aforementioned questions, that is, *to develop a fast subtree kernel that scales up to large, labeled graphs.*

The remainder of this article is structured as follows. In Section 2, we review the subtree kernel from the literature and its runtime complexity. In Section 3, we describe an alternative subtree kernel and its efficient computation based on the Weisfeiler-Lehman test of isomorphism. In Section 4, we compare these two subtree kernels to each other, as well as to a set of four other state-of-the-art graph kernels and report results on kernel computation runtime and classification accuracy on graph benchmark datasets.

## 2 The Ramon-Gärtner subtree kernel

**Terminology**  We define a graph $G$ as a triplet $(V, E, \mathcal{L})$, where $V$ is the set of vertices, $E$ the set of undirected edges, and $\mathcal{L} : V \to \Sigma$ a function that assigns labels from an alphabet $\Sigma$ to nodes in the graph[1]. The neighbourhood $\mathcal{N}(v)$ of a node $v$ is the set of nodes to which $v$ is connected by an edge, that is $\mathcal{N}(v) = \{v'|(v, v') \in E\}$. For simplicity, we assume that every graph has $n$ nodes, $m$ edges, a maximum degree of $d$, and that there are $N$ graphs in our given set of graphs.

A walk is a sequence of nodes in a graph, in which consecutive nodes are connected by an edge. A path is a walk that consists of distinct nodes only. A *(rooted) subtree* is a subgraph of a graph, which has no cycles, but a designated root node. A subtree of G can thus be seen as a connected subset of distinct nodes of G with an underlying tree structure. The height of a subtree is the maximum distance between the root and any other node in the graph plus one. The notion of walk is extending the notion of path by allowing nodes to be equal. Similarly, the notion of subtrees can be extended to *subtree patterns* (also called 'tree-walks' [1]), which can have nodes that are equal. These repetitions of the same node are then treated as distinct nodes, such that the pattern is still a cycle-free tree. Note that all subtree kernels compare subtree *patterns* in two graphs, not (strict) subtrees. Let $S(G)$ refer to the set of all subtree patterns in graph $G$.

**Definition**  The first subtree kernel on graphs was defined by [10]. It compares all pairs of nodes from graphs $G = (V, E, \mathcal{L})$ and $G' = (V', E', \mathcal{L}')$ by iteratively comparing their neighbourhoods:

$$k_{Ramon}^{(h)}(G, G') = \sum_{v \in V} \sum_{v' \in V'} k_h(v, v'), \tag{1}$$

where

$$k_h(v, v') = \left\{ \begin{array}{rl} \delta(\mathcal{L}(v), \mathcal{L}'(v')), & \text{if } h = 1 \\ \lambda_r \lambda_s \sum_{R \in \mathcal{M}(v,v')} \prod_{(w,w') \in R} k_{h-1}(w, w'), & \text{if } h > 1 \end{array} \right. \tag{2}$$

and

$$\mathcal{M}(v, v') = \{R \subseteq \mathcal{N}(v) \times \mathcal{N}(v') | (\forall (u, u'), (w, w') \in R : u = w \Leftrightarrow u' = w')$$
$$\wedge (\forall (u, u') \in R : \mathcal{L}(u) = \mathcal{L}'(u'))\}. \tag{3}$$

Intuitively, $k_{Ramon}$ iteratively compares all matchings $\mathcal{M}(v, v')$ between neighbours of two nodes $v$ from $G$ and $v'$ from $G'$.

**Complexity**  The runtime complexity of the subtree kernel for a pair of graphs is $O(n^2 h 4^d)$, including a comparison of all pairs of nodes $(n^2)$, and a pairwise comparison of all matchings in their neighbourhoods in $O(4^d)$, which is repeated in $h$ iterations. $h$ is a multiplicative factor, not an exponent, as one can implement the subtree kernel recursively, starting with $k_1$ and recursively computing $k_h$ from $k_{h-1}$. For a dataset of $N$ graphs, the resulting runtime complexity is then obviously in $O(N^2 n^2 h 4^d)$.

**Related work**  The subtree kernels in [9] and [1] refine the above definition for applications in chemoinformatics and hand-written digit recognition. Mahé and Vert [9] define extensions of the classic subtree kernel that avoid tottering [8] and consider unbalanced subtrees. Both [9] and [1] propose to consider $\alpha$-ary subtrees with at most $\alpha$ children per node. This restricts the set of matchings to matchings of up to $\alpha$ nodes, but the runtime complexity is still exponential in this parameter $\alpha$, which both papers describe as feasible on small graphs (with approximately 20 nodes) with many distinct node labels. We present a subtree kernel that is efficient to compute on graphs with hundreds and thousands of nodes next.

# 3 Fast subtree kernels

## 3.1 The Weisfeiler-Lehman test of isomorphism

Our algorithm for computing a fast subtree kernel builds upon the Weisfeiler-Lehman test of isomorphism [14], more specifically its 1-dimensional variant, also known as "naive vertex refinement", which we describe in the following.

Assume we are given two graphs $G$ and $G'$ and we would like to test whether they are isomorphic. The 1-dimensional Weisfeiler-Lehman test proceeds in iterations, which we index by $h$ and which comprise the following steps:

---

**Algorithm 1** One iteration of the 1-dimensional Weisfeiler-Lehman test of graph isomorphism

1: Multiset-label determination
- For $h = 1$, set $M_h(v) := l_0(v) = \mathcal{L}(v)$ for labeled graphs, and $M_h(v) := l_0(v) = |\mathcal{N}(v)|$ for unlabeled graphs.
- For $h > 1$, assign a multiset-label $M_h(v)$ to each node $v$ in $G$ and $G'$ which consists of the multiset $\{l_{h-1}(u)|u \in \mathcal{N}(v)\}$.

2: Sorting each multiset
- Sort elements in $M_h(v)$ in ascending order and concatenate them into a string $s_h(v)$.
- Add $l_{h-1}(v)$ as a prefix to $s_h(v)$.

3: Sorting the set of multisets
- Sort all of the strings $s_h(v)$ for all $v$ from $G$ and $G'$ in ascending order.

4: Label compression
- Map each string $s_h(v)$ to a new compressed label, using a function $f : \Sigma^* \to \Sigma$ such that $f(s_h(v)) = f(s_h(w))$ if and only if $s_h(v) = s_h(w)$.

5: Relabeling
- Set $l_h(v) := f(s_h(v))$ for all nodes in $G$ and $G'$.

---

The sorting step 3 allows for a straightforward definition and implementation of $f$ for the compression step 4: one keeps a counter variable for $f$ that records the number of distinct strings that $f$ has compressed before. $f$ assigns the current value of this counter to a string if an identical string has been compressed before, but when one encounters a new string, one increments the counter by one and $f$ assigns its value to the new string. The sorted order from step 3 guarantees that all identical strings are mapped to the same number, because they occur in a consecutive block.

The Weisfeiler-Lehman algorithm terminates after step 5 of iteration $h$ if $\{l_h(v)|v \in V\} \neq \{l_h(v')|v' \in V'\}$, that is, if the sets of newly created labels are not identical in $G$ and $G'$. The graphs are then not isomorphic. If the sets are identical after $n$ iterations, the algorithm stops without giving an answer.

**Complexity**  The runtime complexity of Weisfeiler-Lehman algorithm with $h$ iterations is $O(hm)$. Defining the multisets in step 1 for all nodes is an $O(m)$ operation. Sorting each multiset is an $O(m)$ operation for all nodes. This efficiency can be achieved by using Counting Sort, which is an instance of Bucket Sort, due to the limited range that the elements of the multiset are from. The elements of each multiset are a subset of $\{f(s_h(v))|v \in V\}$. For a fixed $h$, the cardinality of this set is upper-bounded by $n$, which means that we can sort all multisets in $O(m)$ by the following procedure: We assign the elements of all multisets to their corresponding buckets, recording which multiset they came from. By reading through all buckets in ascending order, we can then extract the sorted multisets for all nodes in a graph. The runtime is $O(m)$ as there are $O(m)$ elements in the multisets of a graph in iteration $h$. Sorting the resulting strings is of time complexity $O(m)$ via the Radix Sort. The label compression requires one pass over all strings and their characters, that is $O(m)$. Hence all these steps result in a total runtime of $O(hm)$ for $h$ iterations.

## 3.2 The Weisfeiler-Lehman kernel on pairs of graphs

Based on the Weisfeiler-Lehman algorithm, we define the following kernel function.

**Definition 1** *The Weisfeiler-Lehman kernel on two graphs $G$ and $G'$ is defined as:*

$$k_{WL}^{(h)}(G, G') = |\{(s_i(v), s_i(v')) | f(s_i(v)) = f(s_i(v')), i \in \{1, \ldots, h\}, v \in V, v' \in V'\}|, \quad (4)$$

*where $f$ is injective and the sets $\{f(s_i(v)) | v \in V \cup V'\}$ and $\{f(s_j(v)) | v \in V \cup V'\}$ are disjoint for all $i \neq j$.*

That is, the Weisfeiler-Lehman kernel counts common *multiset strings* in two graphs.

**Theorem 2** *The Weisfeiler-Lehman kernel is positive definite.*

**Proof** Intuitively, $k_{WL}^{(h)}$ is a kernel because it counts matching subtree patterns of up to height $h$ in two graphs. More formally, let us define a mapping $\phi$ that counts the occurrences of a particular label sequence $s$ in $G$ (generated in $h$ iterations of Weisfeiler-Lehman). Let $\phi_s^{(h)}(G)$ denote the number of occurrences of $s$ in $G$, and analogously $\phi_s^{(h)}(G')$ for $G'$. Then

$$\begin{aligned} k_s^{(h)}(G, G') = \phi_s^{(h)}(G)\phi_s^{(h)}(G') = \\ = |\{(s_i(v), s_i(v')) | s_i(v) = s_i(v'), i \in \{1, \ldots, h\}, v \in V, v' \in V'\}|, \quad (5) \end{aligned}$$

and if we sum over all $s$ from $\Sigma^*$, we obtain

$$\begin{aligned} k_{WL}^{(h)}(G, G') = \sum_{s \in \Sigma^*} k_s^{(h)}(G, G') = \sum_{s \in \Sigma^*} \phi_s^{(h)}(G)\phi_s^{(h)}(G') = \\ = |\{(s_i(v), s_i(v')) | s_i(v) = s_i(v'), i \in \{1, \ldots, h\}, v \in V, v' \in V'\}| = \\ = |\{(s_i(v), s_i(v')) | f(s_i(v)) = f(s_i(v')), i \in \{1, \ldots, h\}, v \in V, v' \in V'\}|, \quad (6) \end{aligned}$$

where the last equality follows from the fact that $f$ is injective.

As $f(s) \neq s$ and hence each string $s$ corresponds to exactly one subtree pattern $t$, $k_{WL}^{(h)}$ defines a kernel with corresponding feature map $\phi_{WL}^{(h)}$, such that

$$\phi_{WL}^{(h)}(G) = (\phi_s^{(h)}(G))_{s \in \Sigma^*} = (\phi_t^{(h)}(G))_{t \in S(G)}. \quad (7)$$

∎

**Theorem 3** *The Weisfeiler-Lehman kernel on a pair of graphs $G$ and $G'$ can be computed in $O(hm)$.*

**Proof** This follows directly from the definition of the Weisfeiler-Lehman kernel and the runtime complexity of the Weisfeiler-Lehman test, as described in Section 3.1. The number of matching multiset strings can be counted as part of step 3, as they occur consecutively in the sorted order. ∎

### 3.3 The Weisfeiler-Lehman kernel on N graphs

For computing the Weisfeiler-Lehman kernel on $N$ graphs we propose the following algorithm which improves over the naive, $N^2$-fold application of the kernel from (4). We now process all $N$ graphs simultaneously and conduct the steps given in the Algorithm 2 in each of $h$ iterations on each graph $G$.

The hash function $g$ can be implemented efficiently: it again keeps a counter variable $x$ which counts the number of distinct strings that $g$ has mapped to compressed labels so far. If $g$ is applied to a string that is different from all previous ones, then the string is mapped to $x + 1$, and $x$ increments. As before, $g$ is required to keep sets of compressed labels from different iterations disjoint.

**Theorem 4** *For $N$ graphs, the Weisfeiler-Lehman kernel on all pairs of these graphs can be computed in $O(Nhm + N^2hn)$.*

**Proof** Naive application of the kernel from definition (4) for computing an $N \times N$ kernel matrix would require a runtime of $O(N^2hm)$. One can improve upon this runtime complexity by computing $\phi_{WL}^{(h)}$ explicitly. This can be achieved by replacing the compression mapping $f$ in the classic Weisfeiler-Lehman algorithm by a hash function $g$ that is applied to all $N$ graphs simultaneously.

---
**Algorithm 2** One iteration of the Weisfeiler-Lehman kernel on $N$ graphs
---
1: Multiset-label determination
 - Assign a multiset-label $M_h(v)$ to each node $v$ in $G$ which consists of the multiset $\{l_{h-1}(u)|u \in \mathcal{N}(v)\}$.

2: Sorting each multiset
 - Sort elements in $M_h(v)$ in ascending order and concatenate them into a string $s_h(v)$.
 - Add $l_{h-1}(v)$ as a prefix to $s_h(v)$.

3: Label compression
 - Map each string $s_h(v)$ to a compressed label using a hash function $g : \Sigma^* \to \Sigma$ such that $g(s_h(v)) = g(s_h(w))$ if and only if $s_h(v) = s_h(w)$.

4: Relabeling
 - Set $l_h(v) := g(s_h(v))$ for all nodes in $G$.
---

This has the following effects on the runtime of Weisfeiler-Lehman: Step 1, the multiset-label determination, still requires $O(Nm)$. Step 2, the sorting of the elements in each multiset can be done via a joint Bucket Sort (Counting Sort) of all strings, requiring $O(Nn + Nm)$ time. The use of the hash function $g$ renders the sorting of all strings unnecessary (Step 3 from Section 3.1), as identical strings will be mapped to the same (compressed) label anyway. Step 4 and Step 5 remain unchanged.

The effort of computing $\phi_{WL}^{(h)}$ on all $N$ graphs in $h$ iterations is then $O(Nhm)$, assuming that $m > n$. To get all pairwise kernel values we have to multiply all feature vectors, which requires a runtime of $O(N^2hn)$, as each graph $G$ has at most $hn$ non-zero entries in $\phi_{WL}^{(h)}(G)$. ∎

### 3.4 Link to the Ramon-Gärtner kernel

The Weisfeiler-Lehman kernel can be defined in a recursive fashion which elucidates its relation to the Ramon-Gärtner kernel.

**Theorem 5** *The kernel* $k_{recursive}^{(h)}$ *defined as*

$$k_{recursive}^{(h)}(G, G') = \sum_{i=1}^{h} \sum_{v \in V} \sum_{v' \in V'} k_i(v, v'), \tag{8}$$

*where*

$$k_i(v, v') = \begin{cases} \delta(\mathcal{L}(v), \mathcal{L}'(v')), & \text{if } i = 1 \\ k_{i-1}(v, v') \max_{R \in \mathcal{M}(v,v')} \prod_{(w,w') \in R} k_{i-1}(w, w'), & \text{if } i > 1 \text{ and } \mathcal{M} \neq \emptyset \\ 0, & \text{if } i > 1 \text{ and } \mathcal{M} = \emptyset \end{cases} \tag{9}$$

*and*

$$\mathcal{M}(v, v') = \{R \subseteq \mathcal{N}(v) \times \mathcal{N}(v') | (\forall (u, u'), (w, w') \in R : u = w \Leftrightarrow u' = w')$$
$$\wedge (\forall (u, u') \in R : \mathcal{L}(u) = \mathcal{L}'(u') \wedge |R| = |\mathcal{N}(v)| = |\mathcal{N}(v')|)\} \tag{10}$$

*is equivalent to the Weisfeiler-Lehman kernel* $k_{WL}^{(h)}$.

**Proof** We prove this theorem by induction over $h$. Induction initialisation: $h = 1$:

$$k_{WL}^{(1)} = |\{(s_1(v), s_1(v')) | f(s_1(v)) = f(s_1(v')), v \in V, v' \in V'\}| = \tag{11}$$

$$= \sum_{v \in V} \sum_{v' \in V'} \delta(\mathcal{L}(v), \mathcal{L}'(v')) = k_{recursive}^{(1)}. \tag{12}$$

The equality follows from the definition of $\mathcal{M}(v, v')$.

Induction step $h \to h + 1$: Assume that $k_{WL}^{(h)} = k_{recursive}^{(h)}$. Then

$$k_{recursive}^{(h+1)} = \sum_{v \in V} \sum_{v' \in V'} k_{h+1}(v, v') + \sum_{i=1}^{h} \sum_{v \in V} \sum_{v' \in V'} k_i(v, v') = \tag{13}$$

$$= |\{(s_{h+1}(v), s_{h+1}(v')) | f(s_{h+1}(v)) = f(s_{h+1}(v')), v \in V, v' \in V'\}| + k_{WL}^{(h)} = k_{WL}^{(h+1)}, \tag{14}$$

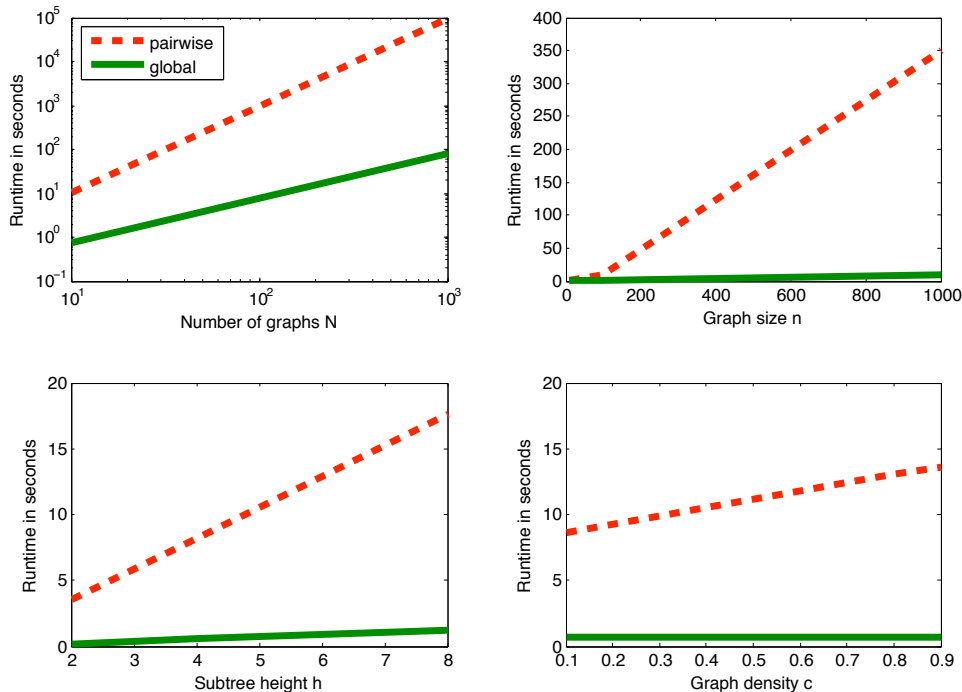

Figure 1: Runtime in seconds for kernel matrix computation on synthetic graphs using the pairwise (red, dashed) and the global (green) Weisfeiler-Lehman kernel (Default values: dataset size $N = 10$, graph size $n = 100$, subtree height $h = 5$, graph density $c = 0.4$).

where the equality of (13) and (14) follows from the fact that $k_{h+1}(v, v') = 1$ if and only if the neigborhoods of $v$ and $v'$ are identical, that is if $f(s_{h+1}(v)) = f(s_{h+1}(v'))$. ∎

Theorem 5 highlights the following differences between the Weisfeiler-Lehman and the Ramon-Gärtner kernel: In (8), Weisfeiler-Lehman considers all subtrees up to height $h$ and the Ramon-Gärtner kernel the subtrees of exactly height $h$. In (9) and (10), the Weisfeiler-Lehman kernel checks whether the neighbourhoods of $v$ and $v'$ match exactly, whereas the Ramon-Gärtner kernel considers all pairs of matching subsets of the neighbourhoods of $v$ and $v'$ in (3). In our experiments, we next examine the empirical differences between these two kernels in terms of runtime and prediction accuracy on classification benchmark datasets.

## 4    Experiments

### 4.1    Runtime behaviour of Weisfeiler-Lehman kernel

**Methods**    We empirically compared the runtime behaviour of our two variants of the Weisfeiler-Lehman (WL) kernel. The first variant computes kernel values pairwise in $O(N^2 hm)$. The second variant computes the kernel values in $O(Nhm + N^2 hn)$ on the dataset simultaneously. We will refer to the former variant as the 'pairwise' WL, and the latter as 'global' WL.

**Experimental setup**    We assessed the behaviour on randomly generated graphs with respect to four parameters: dataset size $N$, graph size $n$, subtree height $h$ and graph density $c$. The density of an undirected graph of $n$ nodes without self-loops is defined as the number of its edges divided by $n(n-1)/2$, the maximal number of edges. We kept 3 out of 4 parameters fixed at their default values and varied the fourth parameter. The default values we used were 10 for $N$, 100 for $n$, 5 for $h$ and 0.4 for the graph density $c$. In more detail, we varied $N$ and $n$ in range $\{10, 100, 1000\}$, $h$ in $\{2, 4, 8\}$ and $c$ in $\{0.1, 0.2, \ldots, 0.9\}$.

For each individual experiment, we generated $N$ graphs with $n$ nodes, and inserted edges randomly until the number of edges reached $\lfloor cn(n-1)/2 \rfloor$. We then computed the pairwise and the global

WL kernel on these synthetic graphs. We report CPU runtimes in seconds in Figure 1, as measured in Matlab R2008a on an Apple MacPro with 3.0GHz Intel 8-Core with 16GB RAM.

**Results**    Empirically, we observe that the pairwise kernel scales quadratically with dataset size $N$. Interestingly, the global kernel scales linearly with $N$. The $N^2$ sparse vector multiplications that have to be performed for kernel computation with global WL do not dominate runtime here. This result on synthetic data indicates that the global WL kernel has attractive scalability properties for large datasets.

When varying the number of nodes $n$ per graph, we observe that the runtime of global WL scales linearly with $n$, and is much faster than the pairwise WL for large graphs.

We observe the same picture for the height $h$ of the subtree patterns. The runtime of both kernels grows linearly with $h$, but the global WL is more efficient in terms of runtime in seconds.

Varying the graph density $c$, both methods show again a linearly increasing runtime, although the runtime of the global WL kernel is close to constant. The density $c$ seems to be a graph property that affects the runtime of the pairwise kernel more severely than that of global WL.

Across all different graph properties, the global WL kernel from Section 3.3 requires less runtime than the pairwise WL kernel from Section 3.2. Hence the global WL kernel is the variant of our Weisfeiler-Lehman kernel that we use in the following graph classification tasks.

## 4.2    Graph classification

**Datasets**    We employed the following datasets in our experiments: MUTAG, NCI1, NCI109, and D&D. MUTAG [3] is a dataset of 188 mutagenic aromatic and heteroaromatic nitro compounds labeled according to whether or not they have a mutagenic effect on the Gram-negative bacterium *Salmonella typhimurium*. We also conducted experiments on two balanced subsets of NCI1 and NCI109, which classify compounds based on whether or not they are active in an anti-cancer screen ([13] and `http://pubchem.ncbi.nlm.nih.gov`). D&D is a dataset of 1178 protein structures [4]. Each protein is represented by a graph, in which the nodes are amino acids and two nodes are connected by an edge if they are less than 6 Angstroms apart. The prediction task is to classify the protein structures into enzymes and non-enzymes.

**Experimental setup**    On these datasets, we compared our Weisfeiler-Lehman kernel to the Ramon-Gärtner kernel ($\lambda_r = \lambda_s = 1$), as well as to several state-of-the-art graph kernels for large graphs: the fast geometric random walk kernel from [12] that counts common labeled walks (with $\lambda$ chosen from the set $\{10^{-2}, 10^{-3}, \ldots, 10^{-6}\}$ by cross-validation on the training set), the graphlet kernel from [11] that counts common induced labeled connected subgraphs of size 3, and the shortest path kernel from [2] that counts pairs of labeled nodes with identical shortest path distance.

We performed 10-fold cross-validation of C-Support Vector Machine Classification, using 9 folds for training and 1 for testing. All parameters of the SVM were optimised on the training dataset only. To exclude random effects of fold assignments, we repeated the whole experiment 10 times. We report average prediction accuracies and standard errors in Tables 1 and 2.

We choose $h$ for our Weisfeiler-Lehman kernel by cross-validation on the training dataset for $h \in \{1, \ldots, 10\}$, which means that we computed 10 different WL kernel matrices in each experiment. We report the total runtime of this computation (*not* the average per kernel matrix).

**Results**    In terms of runtime the Weisfeiler-Lehman kernel can easily scale up even to graphs with thousands of nodes. On D&D, subtree-patterns of height up to 10 were computed in 11 minutes, while no other comparison method could handle this dataset in less than half an hour. The shortest path kernel is competitive to the WL kernel on smaller graphs (MUTAG, NCI1, NCI109), but on D&D its runtime degenerates to more than 23 hours. The Ramon and Gärtner kernel was computable on MUTAG in approximately 40 minutes, but for the large NCI datasets it only finished computation on a subsample of 100 graphs within two days. On D&D, it did not even finish on a subsample of 100 graphs within two days. The random walk kernel is competitive on MUTAG, but as the Ramon-Gärtner kernel, does not finish computation on the full NCI datasets and on D&D within two days. The graphlet kernel is faster than our WL kernel on MUTAG and the NCI datasets, and about a

| Method/Dataset | MUTAG | NCI1 | NCI109 | D & D |
|---|---|---|---|---|
| Weisfeiler-Lehman | 82.05 (±0.36) | 82.19 (± 0.18) | 82.46 (±0.24) | 79.78 (±0.36) |
| Ramon & Gärtner | 85.72 (±0.49) | — | — | — |
| Graphlet count | 75.61 (±0.49) | 66.00 (±0.07) | 66.59 (±0.08) | 78.59 (±0.12) |
| Random walk | 80.72 (±0.38) | — | — | — |
| Shortest path | 87.28 (±0.55) | 73.47 (±0.11) | 73.07 (±0.11) | 78.45 (±0.26) |

—: did not finish in 2 days.

Table 1: Prediction accuracy (± standard error) on graph classification benchmark datasets

| Dataset | MUTAG | NCI1 | | NCI109 | | D & D | |
|---|---|---|---|---|---|---|---|
| Maximum # nodes | 28 | 111 | | 111 | | 5748 | |
| Average # nodes | 17.93 | 29.87 | | 29.68 | | 284.32 | |
| # labels | 7 | 37 | | 54 | | 89 | |
| Number of graphs | 188 | 100 | 4110 | 100 | 4127 | 100 | 1178 |
| Weisfeiler-Lehman | 6” | 5” | 7’20” | 5” | 7’21” | 58” | 11’ |
| Ramon & Gärtner | 40’6” | 25’9” | 29 days* | 26’40” | 31 days* | — | — |
| Graphlet count | 3” | 2” | 1’27” | 2” | 1’27” | 2’40” | 30’21” |
| Random walk | 12” | 58’30” | 68 days* | 2h 9’41” | 153 days* | — | — |
| Shortest path | 2” | 3” | 4’38” | 3” | 4’39” | 58’45” | 23h 17’2” |

—: did not finish in 2 days, * = extrapolated.

Table 2: CPU runtime for kernel computation on graph classification benchmark datasets

factor of 3 slower on D&D. However, this efficiency comes at a price, as the kernel based on size-3 graphlets turns out to lead to poor accuracy levels on three datasets. Using larger graphlets with 4 or 5 nodes that might have been more expressive led to infeasible runtime requirements in initial experiments (not shown here).

On NCI1, NCI109 and D&D, the Weisfeiler-Lehman kernel reached the highest accuracy. On D&D the shortest path and graphlet kernels yielded similarly good results, while on NCI1 and NCI109 the Weisfeiler-Lehman kernel improves by more than 8% the best accuracy attained by other methods. On MUTAG, it reaches the third best accuracy among all methods considered. We could not assess the performance of the Ramon & Gärtner kernel and the random walk kernel on larger datasets, as their computation did not finish in 48 hours. The labeled size-3 graphlet kernel achieves low accuracy levels, except on D&D.

To summarize, the WL kernel turns out to be competitive in terms of runtime on all smaller datasets, fastest on the large protein dataset, and its accuracy levels are highest on three out of four datasets.

# 5 Conclusions

We have defined a fast subtree kernel on graphs that combines scalability with the ability to deal with node labels. It is competitive with state-of-the-art kernels on several classification benchmark datasets in terms of accuracy, even reaching the highest accuracy level on three out of four datasets, and outperforms them significantly in terms of runtime on large graphs, even the efficient computation schemes for random walk kernels [12] and graphlet kernels [11] that were recently defined.

This new kernel opens the door to applications of graph kernels on large graphs in bioinformatics, for instance, protein function prediction via detailed graph models of protein structure on the amino acid level, or on gene networks for phenotype prediction. An exciting algorithmic question for further studies will be to consider kernels on graphs with continuous or high-dimensional node labels and their efficient computation.

# Acknowledgements

The authors would like to thank Kurt Mehlhorn, Pascal Schweitzer, and Erik Jan van Leeuwen for fruitful discussions.

## Footnotes

[1]The extension of this definition and our results to graphs with edge labels is straightforward, but omitted for clarity of presentation.

# References

[1] F. R. Bach. Graph kernels between point clouds. In *ICML*, pages 25–32, 2008.

[2] K. M. Borgwardt and H.-P. Kriegel. Shortest-path kernels on graphs. In *Proc. Intl. Conf. Data Mining*, pages 74–81, 2005.

[3] A. K. Debnath, R. L. Lopez de Compadre, G. Debnath, A. J. Shusterman, and C. Hansch. Structure-activity relationship of mutagenic aromatic and heteroaromatic nitro compounds. correlation with molecular orbital energies and hydrophobicity. *J Med Chem*, 34:786–797, 1991.

[4] P. D. Dobson and A. J. Doig. Distinguishing enzyme structures from non-enzymes without alignments. *J Mol Biol*, 330(4):771–783, Jul 2003.

[5] T. Gärtner, P.A. Flach, and S. Wrobel. On graph kernels: Hardness results and efficient alternatives. In B. Schölkopf and M. Warmuth, editors, *Sixteenth Annual Conference on Computational Learning Theory and Seventh Kernel Workshop, COLT*. Springer, 2003.

[6] T. Horvath, T. Gärtner, and S. Wrobel. Cyclic pattern kernels for predictive graph mining. In *Proceedings of the International Conference on Knowledge Discovery and Data Mining*, 2004.

[7] H. Kashima, K. Tsuda, and A. Inokuchi. Marginalized kernels between labeled graphs. In *Proceedings of the* 20th *International Conference on Machine Learning (ICML)*, Washington, DC, United States, 2003.

[8] P. Mahé, N. Ueda, T. Akutsu, J.-L. Perret, and J.-P. Vert. Extensions of marginalized graph kernels. In *Proceedings of the Twenty-First International Conference on Machine Learning*, 2004.

[9] P. Mahé and J.-P. Vert. Graph kernels based on tree patterns for molecules. *q-bio/0609024*, September 2006.

[10] J. Ramon and T. Gärtner. Expressivity versus efficiency of graph kernels. Technical report, First International Workshop on Mining Graphs, Trees and Sequences (held with ECML/PKDD'03), 2003.

[11] N. Shervashidze, S.V.N. Vishwanathan, T. Petri, K. Mehlhorn, and K. M. Borgwardt. Efficient graphlet kernels for large graph comparison. In *Artificial Intelligence and Statistics*, 2009.

[12] S. V. N. Vishwanathan, Karsten Borgwardt, and Nicol N. Schraudolph. Fast computation of graph kernels. In B. Schölkopf, J. Platt, and T. Hofmann, editors, *Advances in Neural Information Processing Systems 19*, Cambridge MA, 2007. MIT Press.

[13] N. Wale and G. Karypis. Comparison of descriptor spaces for chemical compound retrieval and classification. In *Proc. of ICDM*, pages 678–689, Hong Kong, 2006.

[14] B. Weisfeiler and A. A. Lehman. A reduction of a graph to a canonical form and an algebra arising during this reduction. *Nauchno-Technicheskaya Informatsia, Ser. 2*, 9, 1968.

